# GDS: Gradient Descent Generation of Symbolic Classification Rules

**Reinhard Blasig**
Kaiserslautern University, Germany
Present address: Siemens AG, ZFE ST SN 41
81730 München, Germany

## Abstract

Imagine you have designed a neural network that successfully learns a complex classification task. What are the relevant input features the classifier relies on and how are these features combined to produce the classification decisions? There are applications where a deeper insight into the structure of an adaptive system and thus into the underlying classification problem may well be as important as the system's performance characteristics, e.g. in economics or medicine. GDS[1] is a backpropagation-based training scheme that produces networks transformable into an **equivalent** and **concise** set of IF-THEN rules. This is achieved by imposing penalty terms on the network parameters that adapt the network to the expressive power of this class of rules. Thus during training we simultaneously minimize classification and transformation error. Some real-world tasks demonstrate the viability of our approach.

## 1 Introduction

This paper deals with backpropagation networks trained to perform a classification task on Boolean or real-valued data. Given such a classification task in most cases it is not too difficult to devise a network architecture that is capable of learning the input-output relation as represented by a number of training examples. Once training is finished one has a black box which often does a quite good job not

only on the training patterns but also on some previously unseen test patterns. A good generalization performance indicates that the network has grasped part of the structure inherent in the classification task. The net has figured out which input features are relevant to make a classification decision and which are not. It has also modelled the way the relevant features have to be combined in order to produce the classifying output. In many applications it is important to get an understanding of this information hidden inside the neural network. Not only does this help to create or verify a domain theory, the analysis of this information may also serve human experts to determine, when and in what way the classifier will fail.

In order to explicate the network's implicit information, we transform it into a set of rules. This idea is not new, cf. (Saito and Nakano, 1988), (Bochereau and Bourgine, 1990), (Y. Hayashi, 1991) and (Towell and Shavlik, 1992). In contrast to these approaches, which extract rules after BP-training is finished, we apply penalty terms during training to adapt the network's expressive power to that of the rules we want to generate. Consequently the net will be transformable into an equivalent set of rules.

Due to their good comprehensibility we restrict the rules to be of the form IF $< premise >$ THEN $< conclusion >$, where the premise as well as the conclusion are Boolean expressions. To actually make the transformation two problems have to be solved:

- Neural nets are well known for their distributed representation of information; so in order to transform a net into a concise and comprehensible rule set one has to find a way of condensing this information without substantially changing it.

- In the case of backpropagation networks a continuous activation function determines a node's output depending on its activation. However, the dynamic of this function has no counterpart in the context of rule-based descriptions.

We address these problems by introducing a penalty function $E_P$, which we add to the classification error $E_C$ yielding the total backpropagation error

$$E_T = E_D + \lambda * E_P. \tag{1}$$

## 2    The Penalty Term

The term $E_P$ is intended to have two effects on the network weights. First, by a weight decay component it aims at reducing network complexity by pushing a (hopefully large) fraction of the weights to 0. The smaller the net, the more concise the rules describing its behavior will be. As a positive side effect, this component will tend to act as a form of "Occam's razor": simple networks are more likely to exhibit good generalization than complex ones. Secondly, the penalty term should minimize the error caused by transforming the network into a set of rules. Adopting the common approach that each non-input neuron represents one rule, there would be no transformation error if the neurons' activation function were threshold functions; the Boolean node output would then indicate, whether the conclusion is drawn or not. But since backpropagation neurons use continuous activation functions like

$y = \tanh(x)$ to transform their activation value $x$ into the output value $y$, we are left with the difficulty of interpreting the continuous output of a neuron. Thus our penalty term will be designed to produce a high penalty for those neurons of the backpropagation net, whose behavior cannot be well approximated by threshold neurons, because their activation values are likely to fall into the nonsaturated region of the tanh-function[2].

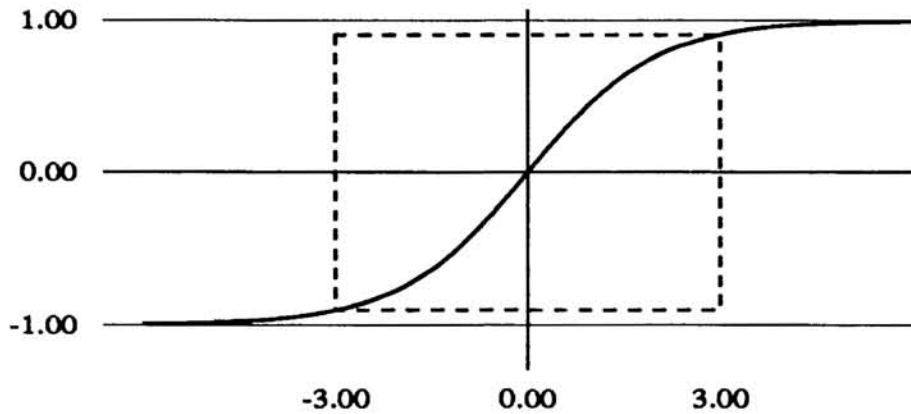

Figure 1: We regard $|x| > 3$ with $|y| = |\tanh(x)| > 0.9$ as the regions, where a sigmoidal neuron can be approximated by a threshold neuron. The nonsaturated region is marked by the dashed box.

For a better understanding of our penalty term one has to be aware of the fact that IF-THEN rules with a Boolean premise and conclusion are essentially Boolean functions. It can easily be shown that any such function can be calculated by a network of threshold neurons provided there is one (sufficiently large) hidden layer. This is still true if we restrict connection weights to the values $\{-1, 0, 1\}$ and node thresholds to be integers (Hertz, Krogh and Palmer, 1991). In order to transfer this scenario to nets with sigmoidal activation functions and having in mind that the activation values of the sigmoidal neurons should always exceed $\pm 3$ (see figure 1), we require the nodes' biases to be odd multiples of $\pm 3$ and the weights $w_{ji}$ to obey

$$w_{ji} \in \{-6, 0, 6\}. \qquad (2)$$

We shortly comment on the practical problem that sometimes bias values as large as $\pm 6m_i$ ($m_i$ being the fan-in of node $i$) may be necessary to implement certain Boolean functions. This may slow down or even block the learning process. A simple solution to this problem is to use some additional input units with a constant output of $+1$. If the connections to these units are also subject to the penalty function $E_P$, it is sufficient to restrict the bias values to

$$b_i \in \{-3, 3\}. \qquad (3)$$

Now we can define penalty functions that push the biases and weights to the desired values. Obviously $E_b$ (the bias penalty) and $E_w$ (the weight penalty) have to be different:

$$E_b(b_i) = |3 - |b_i||  \tag{4}$$

$$E_w(w_{ji}) = \begin{cases} |6 - |w_{ji}|| & \text{for } |w_{ji}| \geq \Theta \\ |w_{ji}| & \text{for } |w_{ji}| < \Theta \end{cases}  \tag{5}$$

The parameter $\Theta$ determines whether a weight should be subject to decay or pushed to attain the value 6 (or $-6$ respectively). Figure 2 displays the graphs of the penalty functions.

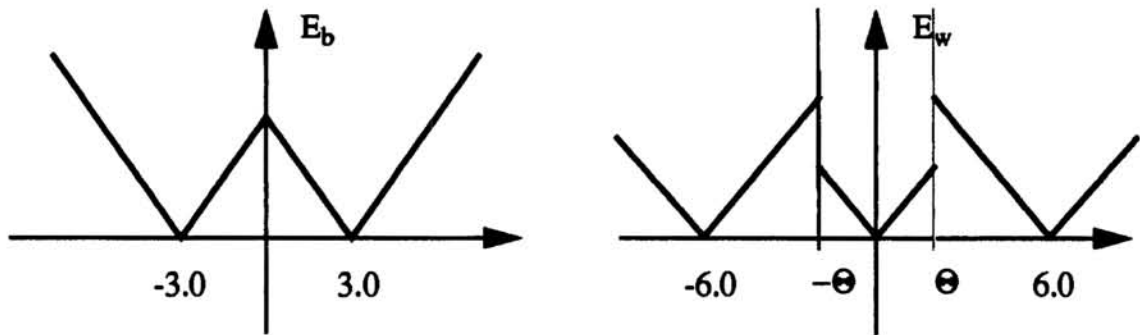

Figure 2: The penalty functions $E_b$ and $E_w$.

The value of $\Theta$ is chosen with the objective that only those weights should exceed this value, which almost certainly have to be nonzero to solve the given classification task. Since we initialize the network with weights uniformly distributed in the interval $[-0.5, 0.5]$, $\Theta = 1.5$ works well at the beginning of the training process. The penalty term then has the effect of a pure weight decay. When learning proceeds and the weights converge, we can slowly reduce the value of $\Theta$, because superfluous weights will already have decayed. So after each sequence of 100 training patterns, say, we decrease $\Theta$ by a factor of 0.995.

Observation shows that weights which once exceeded the value of $\Theta$ quickly reach 6 or $-6$ and that there are relatively few cases where a large weight is reduced again to a value smaller than $\Theta$. Accordingly, the number of weights in $\{-6, 6\}$ successively grows in the course of learning, and the criterion to stop training thus influences the number of nonzero weights.

The end of training is determined by means of cross validation. However, we do not examine the cross validation performance of the trained net, but that of the corresponding rule set. This is accomplished by calculating the performance of the original net with all weights and biases replaced by their optimal values according to (2) and (3).

The weighting factor $\lambda$ of the penalty term (see equation 1) is critical for good learning performance. We pursued the strategy to start learning with $\lambda = 0$, so that the network parameters first move into a region where the classification error is small. If this error falls below a prespecified tolerance level L, $\lambda$ is incremented by 0.001. The factor $\lambda$ goes down by the same amount, when the error grows larger

than $L^3$. By adjusting the weighting factor every 100 training patterns we keep the classification error close to the tolerance level. The choice of L of course depends on the learning task. As a heuristic, L should be slightly larger than the classification error attainable by a non-penalized network.

## 3   Splice-Junction Recognition

The DNA, carrying the genetic information of biological cells, can be thought to be composed of two types of subsequences: exons and introns. The task is to classify each DNA position as either an exon-to-intron transition (EI), an intron-to-exon transition (IE) or neither (N). The only information available is a sequence of 30 nucleotides (A, C, G or T) before and 30 nucleotides after the position to be classified. Splice-junction recognition is a classification task that has already been investigated by a number of machine learning researchers using various adaptive models.

The pattern reservoir contains about 3200 DNA samples, 30% of which were used for training, 10% for cross-validation and 60% for testing. Since we used a grandmother-cell coding for the input DNA sequence, the network has an input layer of 4*60 neurons. With a hidden layer of 20 neurons[4] and two output units for the classes EI and IE, this amounts to about 5000 free parameters. The following table compares the classification performance of our penalty term approach and other machine learning algorithms, cf. (Murphy and Aha, 1992).

Table 1: Splice-junction recognition: error (in percent) of various machine learning algorithms

| algorithm | N | EI | IE | total |
|---|---|---|---|---|
| KBANN | 4.62 | 7.56 | 8.47 | 6.32 |
| GDS | 6.71 | 4.43 | 9.24 | 6.75 |
| Backprop | 5.29 | 5.74 | 10.75 | 6.77 |
| Perceptron | 3.99 | 16.32 | 17.41 | 10.43 |
| ID3 | 8.84 | 10.58 | 13.99 | 10.56 |
| Nearest Neighbor | 31.11 | 11.65 | 9.09 | 20.74 |

Surprisingly, the GDS network turned out to be very small. The weight decay component of our penalty term managed to push all but 61 weights to zero, making use of only three hidden neurons. Thus in addition to performing very well, the network is transformable into a concise rule set, as follows[5]:

```
hidden(2):   at least 4 nucleotides match    sequence @1:  'GTAXG'
hidden(11):  at least 3 nucleotides match    sequence @-3: 'YAG'
hidden(17):  at least 1 nucleotides matches  sequence @-1: 'GG'

class EI: hidden(2) AND hidden(11)
class IE: NOT(hidden(2)) AND hidden(17)
```

## 4   Prediction of Interest Rates

This is an application, where the network input is a vector of real numbers. Since our approach can only handle binary input, we supplement the net with a discretization layer that provides a thermometer code representation (Hancock 1988) of the continuous valued input. In contrast to pure Boolean learning algorithms (Goodman, Miller and Smyth, 1989), (Mezard and Nadal, 1989), which can also be endowed with discretization facilities, here the discretization process is fully integrated into the learning scheme, as the discretization intervals will be adapted by the backpropagation algorithm.

The data comprises a total of 226 patterns, which we distribute randomly on three sets: training set (60%), cross-validation set (20%) and test set (20%). The input represents the monthly development of 14 economic time series during the last 19 years. The Boolean target indicates, whether the interest rates will go up or down during the six months succeeding the reference month[6]. The time series include among others month of the year, income of private households or the amount of German foreign investments. For some time series it is useful not to take the raw feature measurements as input, but the difference between two succeeding measurements; this is advantageous if the underlying time series show only small changes relative to their absolute values. All series were normalized to have values in the range from $-1$ to $+1$.

We used a network containing a discretization layer of two neurons per input dimension. So there are 28 discretization neurons, which are fully connected to the 10 hidden nodes. The output layer consists of a single neuron. Since our data set is relatively small, the intention to obtain simple rules is not only motivated by the objective of comprehensibility, but also by the notion that we cannot expect a large rule set to be justified by a small amount of training data. In fact, during training 90% of the weights were set to zero and three hidden units proved to be sufficient for this task. Nevertheless the prediction error on the test set could be reduced to 25%. This compares to an error rate of about 20% attainable by a standard backpropagation network with one hidden layer of ten neurons and no input discretization. We thus sacrificed 5% of prediction performance to yield a very compact net, that can be easily transformed into a set of rules. Some of the generated rules are shown below. The first rule e.g. states that interest rates will rise if private income increases AND foreign investments decrease by a certain amount during the reference month.

If the rules produce contradicting predictions for a given input, the final decision will be made according to a majority vote. A tie is broken by the bias value of the

output unit, which states that by default interest rates will rise.

```
IF (at least 2 of { increase of private income < 0.73%,
                    decrease of foreign investments < 64 MIO DM })
THEN (interest rates will rise)
ELSE (interest rates will fall).

IF (at least 3 of { increase of business climate estimate < 1.75%,
                    treasury bonds yields (11 month ago) > 7.35%,
                    treasury bonds yields (12 month ago) > 8.2%,
                    increase of foreign investments < 60 MIO DM })
THEN (interest rates will fall)
ELSE (interest rates will rise).
```

## 5  Conclusion and Future Work

GDS is a learning algorithm that utilizes a penalty term in order to prepare a backpropagation network for rule extraction. The term is designed to have two effects on the network's weights:

- By a weight decay component, the number of nonzero weights is reduced: thus we get a net that can hopefully be transformed into a concise and comprehensible rule set.

- The penalty term encourages weight constellations that keep the node activations out of the nonsaturated part of the activation function. This is motivated by the fact that rules of the type IF < *premise* > THEN < *conclusion* > can only mimic the behavior of threshold units.

The important point is that our penalty function adapts the net to the expressive power of the type of rules we wish to obtain. Consequently, we are able to transform the network into an equivalent rule set. The applicability of GDS was demonstrated on two tasks: splice-junction recognition and the prediction of German interest rates. In both cases the generated rules not only showed a generalization performance close to or even superior to what can be attained by other machine learning approaches such as MLPs or ID3. The rules also prove to be very concise and comprehensible. This is even more remarkable, since both applications represent real-world tasks with a large number of inputs.

Clearly the applied penalty terms impose severe restrictions on the network parameters: besides minimizing the number of nonzero weights, the weights are restricted to a small set of distinct values. Last but not least, the simplification of sigmoidal to threshold units also affects the net's computational power. There are applications, where such a strong bias may negatively influence the net's learning capabilities. Furthermore our current approach is only applicable to tasks with binary target patterns. These limitations can be overcome by dealing with more general rules than those of the Boolean IF-THEN type. Future work will go into this direction.

## Acknowledgements

I wish to thank Hans-Georg Zimmermann and Ferdinand Hergert for many useful discussions and for providing the data on interest rates, and Patrick Murphy and David Aha for providing the UCI Repository of ML databases. This work was supported by a grant of the Siemens AG, Munich.

## Footnotes

[1]Gradient Descent Symbolic Rule Generation

[2]We have to point out that the conversion of sigmoidal neurons to threshold neurons will reduce the net's computational power: there are Boolean functions which can be computed by a net of sigmoidal neurons, but which exceed the capacity of a threshold net of the same topology (Maass, Schnitger and Sontag, 1991). Note that the objective to use threshold units is a consequence of the decision to search for rules of the type IF $< premise >$ THEN $< conclusion >$. A failure of the net to simultaneously minimize both parts of the error measure may indicate that other rule types are more adequate to handle the given classification task.

[3]Negative $\lambda$-values are not allowed.

[4]A reasonable size, considering the experiments described in (Shavlik et al., 1991)

[5]We adopt a notation commonly used in this domain: @n denotes the position of the first nucleotide in the given sequence being left (negative $n$) or right (positive $n$) to the point to be classified. Nucleotide 'Y' stands for ('C' or 'T'), 'X' is any of $\{A, C, G, T\}$. Consequently, e.g. neuron *hidden*(2) is active iff at least four of the five nucleotides of the sequence 'GTAXG' are identical to the input pattern at positions 1 to 5 right of the possible splice junction.

[6]I.e. the month where the input data has been measured.

## References

L. Bochereau, P. Bourgine. (1990) Extraction of Semantic Features and Logical Rules from a Multilayer Neural Network. *Proceedings of the 1990 IJCNN - Washington DC*, Vol.II 579-582.

R.M. Goodman, J.W. Miller, P. Smyth. (1989) An Information Theoretic Approach to Rule-Based Connectionist Expert Systems. *Advances in Neural Information Processing Systems 1*, 256-263. San Mateo, CA: Morgan Kaufmann.

P.J.B. Hancock. (1988) Data Representation in Neural Nets: an Empirical Study. *Proc. Connectionist Summer School.*

Y. Hayashi. (1991) A Neural Expert System with Automated Extraction of Fuzzy If-Then Rules and its Application to Medical Diagnosis. *Advances in Neural Information Processing Systems 3*, 578-584. San Mateo, CA: Morgan Kaufmann.

J. Hertz, A. Krogh, R.G. Palmer. (1991) Introduction to the Theory of Neural Computation. Addison-Wesley.

C.M. Higgins, R.M. Goodman. (1991) Incremental Learning with Rule-Based Neural Networks. *Proceedings of the 1991 IEEE INNS International Joint Conference on Neural Networks - Seattle*, Vol.I 875-880.

M. Mezard, J.-P. Nadal. (1989) Learning in Feedforward Layered Networks: The Tiling Algorithm. *J. Phys. A: Math. Gen. 22*, 2191-2203.

W. Maass, G. Schnitger, E.D. Sontag. (1991) On the Computational Power of Sigmoids versus Boolean Threshold Circuits. *Proceedings of the 32nd Annual IEEE Symposium on Foundations of Computer Science*, 767-776.

P.M. Murphy, D.W. Aha. (1992). *UCI Repository of machine learning databases* [ftp-site: ics.uci.edu: pub/machine-learning-databases]. Irvine, CA: University of California, Department of Information and Computer Science.

J.R. Quinlan. (1986) Induction of Decision Trees. *Machine Learning*, 1: 81-106.

K. Siato, R. Nakano. (1988) Medical diagnostic expert systems based on PDP model. *Proc. IEEE International Conference on Neural Networks* Vol. I 255-262.

V. Tresp, J. Hollatz, S. Ahmad. (1993) Network Structuring and Training Using Rule-Based Knowledge. *Advances in Neural Information Processing Systems 5*, 871-878. San Mateo, CA: Morgan Kaufman.

G.G. Towell, J.W. Shavlik. (1991) Training Knowledge-Based Neural Networks to Recognize Genes in DNA Sequences. In: Lippmann, Moody, Touretzky (eds.), *Advances in Neural Information Processing Systems 3*, 530-536. San Mateo, CA: Morgan Kaufmann.